# A theory of neural integration in the head-direction system

**Richard H.R. Hahnloser[1], Xiaohui Xie and H. Sebastian Seung[1]**
[1]Howard Hughes Medical Institute
Dept. of Brain and Cognitive Sciences
Massachusetts Institute of Technology
Cambridge, MA 02139
{rhahnloser|xhxie|seung}@mit.edu

## Abstract

Integration in the head-direction system is a computation by which horizontal angular head velocity signals from the vestibular nuclei are integrated to yield a neural representation of head direction. In the thalamus, the postsubiculum and the mammillary nuclei, the head-direction representation has the form of a place code: neurons have a preferred head direction in which their firing is maximal [Blair and Sharp, 1995, Blair et al., 1998, **?**].

Integration is a difficult computation, given that head-velocities can vary over a large range. Previous models of the head-direction system relied on the assumption that the integration is achieved in a firing-rate-based attractor network with a ring structure. In order to correctly integrate head-velocity signals during high-speed head rotations, very fast synaptic dynamics had to be assumed.

Here we address the question whether integration in the head-direction system is possible with slow synapses, for example excitatory NMDA and inhibitory GABA(B) type synapses. For neural networks with such slow synapses, rate-based dynamics are a good approximation of spiking neurons [Ermentrout, 1994]. We find that correct integration during high-speed head rotations imposes strong constraints on possible network architectures.

## 1 Introduction

Several network models have been designed to emulate the properties of head-direction neurons (HDNs) [Zhang, 1996, Redish et al., 1996, Goodridge and Touretzky, 2000]. The model by Zhang reproduces persistent activity during stationary head positions. Persistent neural activity is generated in a ring-attractor network with symmetric excitatory and inhibitory synaptic connections. Independently, he and Redish et al. showed that integration is possible by adding asymmetrical connections to the attractor network. They assumed that the strength of these asymmetrical connections is modulated by head-velocity. When the rat moves its head to the right, the asymmetrical connections induce a rightward shift of the activity in the attractor network. A more plausible model without multiplicative

modulation of connections has been studied recently by Goodridge and Touretzky. There, the head-velocity input has a modulatory influence on firing rates of intermittent neurons rather than on connection strengths. The intermittent neurons are divided into two groups that make spatially offset connections, one group to the right, the other to the left. The different types of neurons in the Goodridge and Touretzky model have firing properties that are comparable to neurons in the various nuclei of the head-direction system.

What all these previous models have in common is that the integration is performed in an inherent double-ring network with very fast synapses (less than 1ms for [Goodridge and Touretzky, 2000]). The connections made by one ring are responsible for rightward turns and the connections made by the other ring are responsible for leftward turns. In order to derive a network theory of integration valid for fast *and* slow synapses, here we solve a simple double-ring network in the linear and in the saturated regimes.

An important property of the head-direction system is that the integration be linear over a large range of head-velocities. We are interested in finding those type of synaptic connections that yield a large linear range and pose our findings as predictions on optimal network architectures. Although our network is conceptually simpler than previous models, we show that using two simple read-out methods, averaging and extracting the maximum, it is possible to approximate head-velocity independent tuning curves as observed in the Postsubiculum (PoS) and anticipatory responses in the anterior dorsal thalamus (ADN).

## 2  Definition of the model

We assume that the number of neurons in the double-ring network is large and write its dynamics as a continuous neural field

$$\tau \frac{d}{dt} s_l(\theta, t) + s_l(\theta, t) = f_l(\theta, t) \tag{1}$$

$$\tau \frac{d}{dt} s_r(\theta, t) + s_r(\theta, t) = f_r(\theta, t), \tag{2}$$

where

$$f_l = \left[ \int_{-\pi}^{\pi} \frac{d\theta'}{2\pi} [W_S(\theta - \theta' - \Phi) s_l(\theta', t) + W_D(\theta - \theta' + \Psi) s_r(\theta', t)] + b_0 - \Delta b \right]^+$$

$$f_r = \left[ \int_{-\pi}^{\pi} \frac{d\theta'}{2\pi} [W_D(\theta - \theta' - \Psi) s_l(\theta', t) + W_S(\theta - \theta' + \Phi) s_r(\theta', t)] + b_0 + \Delta b \right]^+.$$

$[x]^+ = \max(0, x)$ denotes a rectification nonlinearity. $f_l(\theta)$ and $f_r(\theta)$ are the firing rates of neurons in the left and right ring, respectively. The quantities $s_l$ and $s_r$ represent synaptic activations (amount of neurotransmitter release caused by the firing rates $f_l$ and $f_r$). $\tau$ is a synaptic time constant. The vestibular inputs $b_0 - \Delta b$ and $b_0 + \Delta b$ are purely excitatory, $-b_0 \leq \Delta b \leq b_0$. For simplicity, we assume that $\Delta b$ is proportional to angular head-velocity. The synaptic connection profiles $W_S$ between neurons on the same ring and $W_D$ between neurons on different rings are given by:

$$W_S(\theta) = J_0 + J_1 \cos(\theta) \qquad W_D(\theta) = K_0 + K_1 \cos(\theta). \tag{3}$$

$J_0$, $J_1$, $K_0$ and $K_1$ define the intra and inter-ring connection strengths. $\Phi$ is the intra-ring connection offset and $\Psi$ the inter-ring offset.

## 3  Integration

When the animal is not moving, the vestibular inputs to the two rings are equal, $\Delta b = 0$. In this case, within a certain range of synaptic connections, steady bumps of activities appear

on the two rings. When the head of the animal rotates, the activity bumps travel at a velocity $v$ determined by $\Delta b$. For perfect integration, $v$ should be proportional to $\Delta b$ over the full range of possible head-velocities. This is a difficult computational problem, in particular for slow synapses.

## 4 Small head-velocity approximation

When the head is not rotating ($\Delta b = 0$), the two stationary bumps of synaptic activation are of the form

$$s_l^*(\theta) = [A\cos(\theta - \theta_0) - C]^+ \quad \text{and} \quad s_r^*(\theta) = [A\cos(\theta - \theta_0 + \beta) - C]^+ \qquad (4)$$

where $\theta_0$ is the current head direction and $\beta$ is the offset between the two bumps. How to calculate $A$, $C$ and $\beta$ is shown in the Appendix. The half width of these bumps is given by

$$\theta_c = \arccos(C/A). \qquad (5)$$

When the angular head velocity is small ($\Delta b/b_0 \ll 1$), we linearize the dynamics around the stationary solution Eq. (4), see Appendix. We find that

$$s_l(\theta - vt) = [(A - \delta A)\cos(\theta - \theta_0 - vt) - (C - \delta C)]^+ \qquad (6)$$
$$s_r(\theta - vt) = [(A + \delta A)\cos(\theta - \theta_0 - vt + \beta) - (C + \delta C)]^+, \qquad (7)$$

where the velocity $v$ is given by

$$\frac{v}{\Delta b} = \frac{2k_1 J_1 \sin(\Phi)}{\pi^2 \tau A}[(k_2(\theta_c(J_0 - K_0) - \pi) - (J_0 - K_0)\sin(\theta_c)], \qquad (8)$$

and

$$k_1 = \frac{1}{J_1\cos\Phi - \sqrt{K_1^2 - J_1^2\sin\Phi}} \qquad (9)$$

$$k_2 = \frac{\theta_C}{2\sin\theta_C} + \frac{\cos\theta_C}{2} - \frac{\pi k_1}{\sin\theta_C}. \qquad (10)$$

Equation (8) is the desired result, relating the velocity $v$ of the two bumps to the differential vestibular input $\Delta b$. In Fig. 1 we show simulation results using slow synapses ($\tau = 80\text{ms}$). The integration is linear over almost the entire range of head-velocities (up to more than $600^\circ/sec$) when $K_1 = J_1$, i.e., when the amplitudes of inter-ring and intra-ring connections are equal. We point out that the condition $K_1 = J_1$ cannot directly be deduced from the above formulas, some empirical tuning (for example $K_0 = 0$) was necessary to achieve this large range of linearity (large both in $\Delta b$ and $v$).

When the bumps move, their amplitudes tend to decrease. Fig. 1d shows the peak firing rates of neurons in the two rings as a function of vestibular input. As can be seen, the firing rates are a linear function of vestibular input, in agreement with equations 17 and 18 of the Appendix. However, a linear firing-rate modulation by head velocity is not universal, for some parameters we have seen asymmetrically head-velocity tuning, with a preference for small head velocities (not shown).

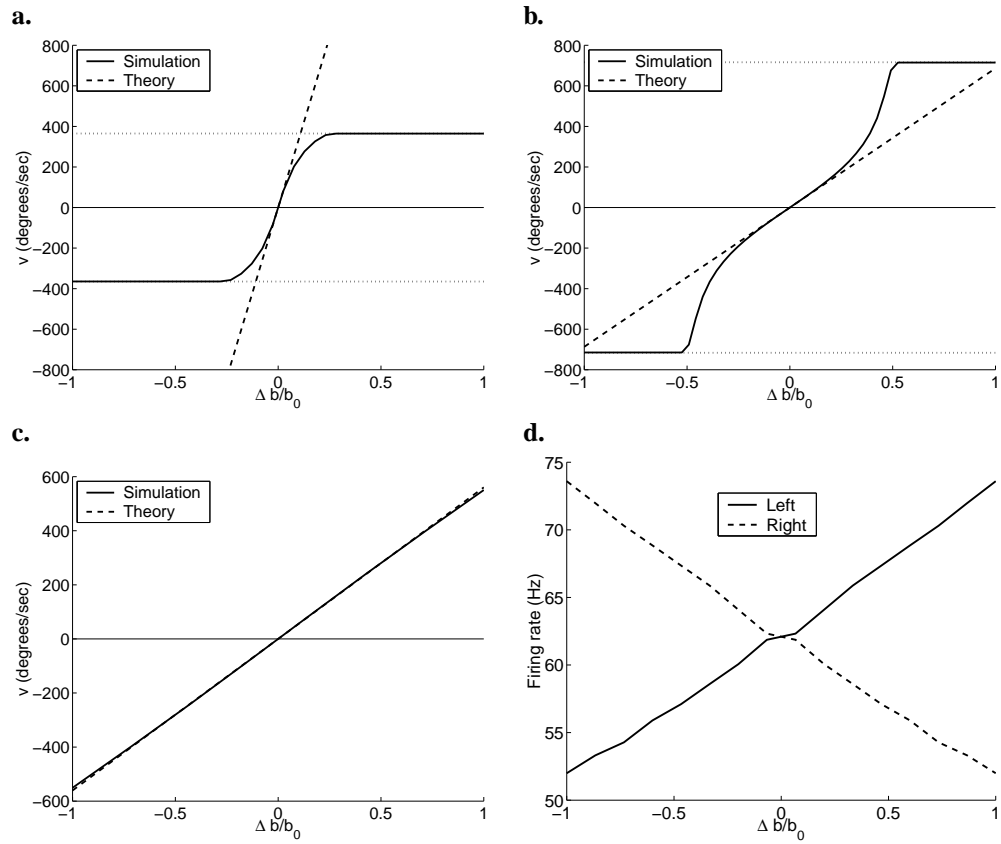

Figure 1: Velocity of activity bumps as a function of vestibular input $\Delta b/b_0$. **a.** Sublinear integration. $K_1 < J_1$, $\Phi = 30°$, $K_0 < 0$. **b.** Supralinear integration. $K_1 > J_1$, $\Phi = 45°$, $K_0 < 0$. **c.** Linear (perfect) integration. $K_1 = J_1$, $\Phi = 72°$, $K_0 = 0$. **d.** Head-velocity dependent modulation of firing rates (on the right and on the left ring). Same parameters as in c. $\tau = 80$ms. $\Phi = 81°$, and $\Psi = 81°$.

## 5   Saturating velocity

When $\Delta b$ is very large, at some point, the left ring becomes inactive. Because inactivating the left ring means that the push-pull competition between the two rings is minimized, we are able to determine the saturating velocity of the double-ring network. The saturating velocity is given by the on-ring connections $W_S$. Define

$$
\begin{aligned}
W(\theta) &= J_0 + J_1 \cos(\theta - \Phi) \\
&= J_0 + J_1 \cos(\Phi)(\cos(\theta) + \tan(\Phi)\sin(\theta)) \\
&= \tilde{W}_S(\theta) - \tan(\Phi)\,\tilde{W}_S'(\theta),
\end{aligned}
$$

where $\tilde{W}_S(\theta) = J_0 + J_1 \cos(\Phi)\cos(\theta)$. Now, let $f^*(\theta)$ be the steady solution of a ring network with symmetric connections $\tilde{W}_S(\theta)$. By differentiating, it follows that $f^*(\theta - \frac{\tan(\Phi)}{\tau} t)$ is the solution of a ring network with connections $W(\theta)$. Hence, the saturating velocity $v_{sat}$ is given by

$$
v_{sat} = \frac{\tan(\Phi)}{\tau}. \tag{11}
$$

Notice that a traveling solution may not always exist if one ring is inactive (this is the case when there are no intra-ring excitatory connections). However, even without a traveling solution, equation (11) remains valid. In Figs. 1a and b, the saturating velocity is indicated by the horizontal dotted lines, in Fig. 1a we find $v_{sat} = 380^\circ/sec$ and in Fig. 1b $v_{sat} = 730^\circ/sec$.

## 6   ADN and POs neurons

Goodridge and Touretzky's integrator model was designed to emulate details of neuronal tuning as observed in the different areas of the head-direction system. Wondering whether the simple double ring studied here can also reproduce multiple tuning curves, we analyze simple read-out methods of the firing rates $f_l$ and $f_r$. What we find is that two read-out methods can indeed approximate response behavior resembling that of ADN and POs neurons.

ADN neurons: By reading out firing rates using a maximum operation, $z(\theta) = \max(f_r(\theta), f_l(\theta))$, anticipatory head-direction tuning arises due to the fact that there is an activity offset $\beta$ between the two rings, equation (13). When the head turns to the right, the activity on the right ring is larger than on the left ring and so the tuning of $z(\theta)$ is biased to the right. Similarly, for left turns, $z(\theta)$ is biased to the left. Thus, the activity offset between the two rings leads to an anticipation time $T$ for ADN neurons, see Figure 2. Because, by assumption $\beta$ is head-velocity independent, it follows that $T$ is inversely proportional to head-velocity (assuming perfect integration), $T = \beta/v/2$. In other words, the anticipation time tends to be smaller for fast head rotations and larger for slow head rotations.

POs neurons: By reading out the double ring activity as an average, $y(\theta) = 1/2(f_r(\theta) + f_l(\theta))$, neurons in POs do not have any anticipation time: because averaging is a symmetric operation, all information about the direction of head rotations is lost.

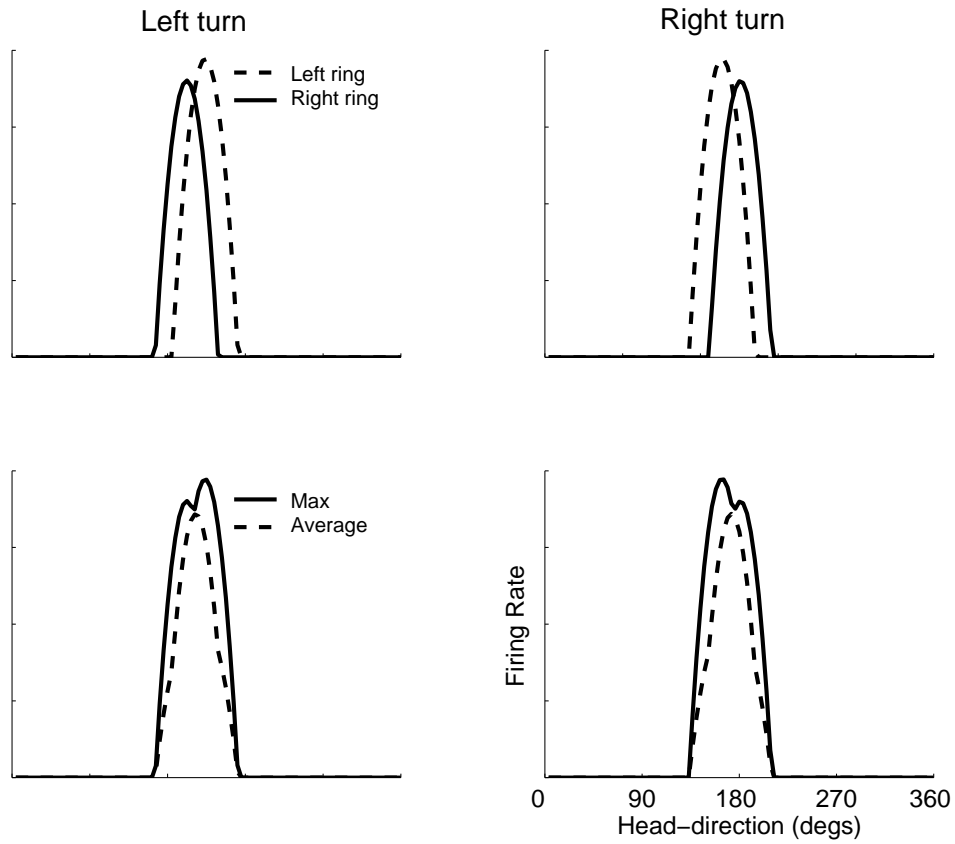

Figure 2: Snapshots of the activities on the two rings (top). Reading out the activities by averaging and by a maximum operation (bottom).

## 7 Discussion

Here we discuss how the various connection parameters contribute to the double-ring network to function as an integrator. In particular we discuss how parameters have to be tuned in order to yield an integration that is large in $\Delta b$ and in $v$.

- $\tau$: By assumption the synaptic time constant $\tau$ is large. $\tau$ has the simplest effect of all parameters on the integrator properties. According to equation (8), $\tau$ scales the range of $v$. Notice that if $\tau$ were small, a large range of $v$ could be trivially achieved. The art here is to achieve this with large $\tau$.

- $\Phi$: The connection offset $\Phi$ between neurons receiving similar vestibular input is the sole parameter besides $\tau$ determing the saturating head-velocity, beyond which integration is impossible. According to equation (11), the saturating velocity is large if $\Phi$ is close to $90^{\circ}$ (we want the saturating velocity to be large). In other words, for good integration, excitatory connections should be strongest (or inhibitory connections weakest) for neuron pairs with preferred head-directions differing by close to $90^{\circ}$.

- $\Psi$: The connection offset $\Psi$ between neurons receiving different vestibular input determines the anticipation time $T$ of thalamic neurons. If $\Psi$ is large, then $\beta$,

the activity offset in equation (13) is large. And, because $\beta$ is proportional to $T$ (assuming perfect integration), we conclude that $\Psi$ should preferentially be large (close to $90°$) if $T$ is to be large. Notice that by equation (8), the range of $v$ is not affected by $\Psi$.

- $K_0$ and $K_1$: The inter-ring connections should be mainly excitatory, which implies that $K_0$ should not be too negative ($K_0 = 0$ was found to be optimal). The intuitive reason is the following. We want the integration to be as linear in $\Delta b$ as possible, which means that we want our linear expansions (6) and (7) to deviate as little as possible from (4). Hence, the differential gain between the two rings should be small, which is the case when the two rings excite each other. The inter-ring excitation makes sure, even for large values of $\Delta b$, that there are comparable activity levels on the two rings. This is one of the main points of this study.

- $J_0$ and $J_1$: The intra-ring connections should be mainly inhibitory, which implies that $J_0$ should be strongly negative. The reason for this is that inhibition is necessary to result in proper and stable integration. Since inhibition cannot come from the inter-ring connections, it has to come from $J_0$. Notice also that according to equation (15), $J_1$ cannot be much larger than $K_1$. If this were the case, the persistent activity in the no head-movement case would become unstable. For linear integration we have found that the condition $K_1 = J_1$ is necessary; small deviations from this condition cause the integrator to become sub- or supralinear.

## 8  Conclusion

We have presented a theory for integration in the head-direction system with slow synapses. We have found that in order to achieve a large range of linear integration, there should be strong excitatory connections between neurons with dissimilar head-velocity tuning and inhibitory connections between neurons with similar head-velocity tuning (see the discussion). Similar to models of the occulomotor integrator [Seung, 1996], we have found that linear integration can only be achieved by precise tuning of synaptic weights (for example $K_1 = J_1$).

## Appendix

To study the traveling pulse solution with velocity $v$, it is convenient to go into a moving coordinate frame by the change of variables $\alpha = \theta - vt$. The stationary solution in the moving frame reads

$$-\tau v s_l'(\alpha) + s_l(\alpha) = f_l(\alpha) \quad \text{and} \quad -\tau v s_r'(\alpha) + s_r(\alpha) = f_r(\alpha) \tag{12}$$

Set $v = 0$. In order to find the fixed points of equation (12), we use the ansatz (4) and equate the coefficients of the 3 Fourier modes $\sin(\alpha)$, $\cos(\alpha)$ and the $\alpha$-independent mode. This leads to

$$\beta = \arcsin(J_1 \sin(\Phi)/K_1) - \Psi \tag{13}$$

$$A = \frac{b_0}{-2(J_0 + K_0)f_0(\theta_c) - \cos(\theta_c)} \tag{14}$$

$$1 = f_2(\theta_c) \left[ J_1 \cos(\Phi) + \sqrt{K_1^2 - J_1^2 \sin^2(\Phi)} \right] \tag{15}$$

where the functions $f_0$ and $f_2$ are given by

$$f_0(\theta_c) = \frac{1}{2\pi}[\sin(\theta_c) - \theta_c \cos(\theta_c)], \quad f_2(\theta_c) = \frac{1}{2\pi}[\theta_c - \frac{1}{2}\sin(2\theta_c)].$$

The above set of equations fully characterize the solution for $v = 0$. Eq. (13) determines the offset $\beta$ between the two rings, eq. (15) determines the threshold $\theta_c$, eq. (14) the amplitude $A$ and eq. (5) the bias $C$.

When the vestibular input $\Delta b$ is small, we assume that the perturbed solution around $s_l^*$ and $s_r^*$ takes the form:

$$
\begin{aligned}
s_l(\theta) &= (A + \delta A_l)\cos(\theta - \theta_0) - (C + \delta C_l) \\
s_r(\theta) &= (A + \delta A_r)\cos(\theta + \beta - \theta_0) - (C + \delta C_r).
\end{aligned}
$$

We linearize the dynamics (12) (to first order in $\theta_c$) and equate the Fourier coefficients. This leads to

$$
v = \frac{J_1 \sin(\Phi)}{2\pi\tau A}[(\theta_c + \sin(2\theta_c)/2)\delta A - 2\sin(\theta_c)\delta C] \tag{16}
$$

where $\delta A \equiv \delta A_r - \delta A_l$ and $\delta C \equiv \delta C_r - \delta C_l$. We determine $\delta A$ and $\delta C$ by solving the linearized dynamics of the differential mode $\delta m = s_r(\theta - \beta) - s_l(\theta) = \delta A\cos(\theta - \theta_0) - \delta C$. Comparing once more the Fourier coefficients leads to

$$
\begin{aligned}
\delta A &= 2\Delta b[(k_3\theta_c - 1)k_2 - k_3\sin(\theta_c)]^{-1} \tag{17} \\
\delta C &= 2k_2\Delta b[(k_3\theta_c - 1)k_2 - k_3\sin(\theta_c)]^{-1}, \tag{18}
\end{aligned}
$$

where $k_3 = (J_0 - K_0)/\pi$. By substituting $\delta A$ and $\delta C$ into Eq. (16), we find equation (8).

# References

[Blair et al., 1998] Blair, H., Cho, J., and Sharp, P. (1998). Role of the lateral mammillary nucleus in the rat head direction circuit: A combined single unit recording and lesion study. *Neuron*, 21:1387–1397.

[Blair and Sharp, 1995] Blair, H. and Sharp, P. (1995). Anticipatory head diirection signals in anterior thalamus: evidence for a thalamocortical circuit that integrates angular head motion to compute head direction. *The Journal of Neuroscience*, 15(9):6260–6270.

[Ermentrout, 1994] Ermentrout, B. (1994). Reduction of conductance-based models with slow synapses to neural nets. *Neural Computation*, 6:679–695.

[Goodridge and Touretzky, 2000] Goodridge, J. and Touretzky, D. (2000). Modeling attractor deformation in the rodent head-direction system. *The Journal of Neurophysiology*, 83:3402–3410.

[Redish et al., 1996] Redish, A., Elga, A. N., and Touretzky, D. (1996). A coupled attractor model of the rodent head direction system. *Network: Computation in Neural Systems*, 7:671–685.

[Seung, 1996] Seung, H. S. (1996). How the brain keeps the eyes still. *Proc. Natl. Acad. Sci. USA*, 93:13339–13344.

[Zhang, 1996] Zhang, K. (1996). Representation of spatial orientation by the intrinsic dynamics of the head-direction cell ensemble: A theory. *J. Neurosci.*, 16(6):2112–2126.
